# Flight Control in the Dragonfly: A Neurobiological Simulation

**William E. Faller** and **Marvin W. Luttges**
Aerospace Engineering Sciences,
University of Colorado, Boulder, Colorado 80309-0429.

## ABSTRACT

Neural network simulations of the dragonfly flight neurocontrol system have been developed to understand how this insect uses complex, unsteady aerodynamics. The simulation networks account for the ganglionic spatial distribution of cells as well as the physiologic operating range and the stochastic cellular firing history of each neuron. In addition the motor neuron firing patterns, "flight command sequences", were utilized. Simulation training was targeted against both the cellular and flight motor neuron firing patterns. The trained networks accurately resynthesized the intraganglionic cellular firing patterns. These in turn controlled the motor neuron firing patterns that drive wing musculature during flight. Such networks provide both neurobiological analysis tools and first generation controls for the use of "unsteady" aerodynamics.

## 1 INTRODUCTION

Hebb (1949) proposed a theory of inter-neuronal learning, "Hebbian Learning", in which cells acting together as assemblies alter the efficacy of mutual interconnections. These neural "cell assemblies" presumably comprise the information processing "units" of the nervous system.

To provide one framework within which to perform detailed analyses of these cellular organizational "rules" a new analytical technique based on neural networks is being explored. The neurobiological data analyzed was obtained from the neural cells of the dragonfly ganglia.

The dragonfly use of unsteady separated flows to generate highly maneuverable flight is governed by the control sequences that originate in the thoracic ganglia flight motor neurons (MN). To provide this control the roughly 2200 cells of the meso- and metathoracic ganglia integrate environmental cues that include visual input, wind shear, velocity and acceleration. The cellular firing patterns coupled with proprioceptive feedback in turn drive elevator/depressor flight MNs which typically produce a 25-37 Hz wingbeat depending on the flight mode (Luttges 1989; Kliss 1989).

The neural networks utilized in the analyses incorporate the spatial distribution of cells, the physiologic operating range of each neuron and the stochastic history of the cellular spike trains (Faller and Luttges 1990). The present work describes two neural networks. The simultaneous single-unit firing patterns at time (t) were used to predict the cellular firing patterns at time (t+$\Delta$). And, the simultaneous single-unit firing patterns were used to "drive" flight-MN firing patterns at a 37 Hz wingbeat frequency.

## 2 METHODS

### 2.1 BIOLOGICAL DATA

Recordings were obtained from the mesothoracic ganglion of the dragonfly *Aeshna* in the ganglionic regions known to contain the cell bodies of flight MNs as well as small and large cell bodies (Simmons 1977; Kliss 1989). Multiple-unit recordings from many cells (~40-80) were systematically decomposed to yield simultaneously active single-unit firing patterns. The technique has been described elsewhere (Faller and Luttges in press).

During the recording of neural activity spontaneous flight episodes commonly occurred. These events were consistent with typical flight episodes (2-3 secs duration) observed in the tethered dragonfly (Somps and Luttges 1985). For analysis, a 12 second record was obtained from 58 single units, 26 rostral cells and 32 caudal cells. The continuous record was separated into 4 second behavioral epochs: pre-flight, flight and post-flight.

A simplified model of one flight mode was assumed. Each forewing is driven by 3 main elevator and 2 main depressor muscles, innervated by 11 and 14 MNs, respectively. A 37 Hz MN firing frequency, 3-5 spikes per output burst, and 180 degree phase shift between antagonistic MNs was assumed. Given the symmetrical nature of the elevator/depressor output patterns only the 11 elevator MNs were simulated.

Prior to analysis the ganglionic spatial distribution of neurons was reconstructed. The importance of this is reserved for later discussion. A method has been described (Faller and Luttges submitted:a) that resolves the spatial distribution based on two distancing criteria: the amplitude ratio across electrodes and the spike angle (width) for each cell. Cells were sorted along a rostral, cell 1, to caudal, cell 58 continuum based on this information.

The middle 2 seconds of the flight data was simulated. This was consistent with the known duration of spontaneous flight episodes. Within these 2 seconds, 44 cells remained active, 19 rostral and 25 caudal. The cell numbering (1-58) derived for the biological data was not altered. The remaining 14 inactive cells/units carry zeros in all analyses.

## 2.2 MIMICKING THE SINGLE CELLS

Each neuron was represented by a unique unit that mimicked both the mean firing frequency and dynamic range of the physiologic cell. The activation value ranged from zero to twice the normalized mean firing frequency for each cell. The dynamic range was calculated as a unique thermodynamic profile for each sigmoidal activation function. The technique has been described fully elsewhere (Faller and Luttges 1990).

## 2.3 SPIKE TRAIN REPRESENTATION

The spike trains and MN firing patterns were represented as iteratively continuous "analog" gradients (Faller and Luttges 1990 & submitted:b). Briefly, each spike train was represented in two-dimensions based on the following assumptions: (1) the mean firing frequency reflects the inherent physiology of each cell and (2) the interspike intervals encode the information transferred to other cells. Exponential functions were mapped into the intervals between consecutive spikes and these functions were then discretized to provide the spike train inputs to the neural network. These functions retain the exact spiking times and the temporal modulations (interval code) of cell firing histories.

## 2.4 ARCHITECTURE

The two simulation architectures were as follows:

|              | Simulation 1                  | Simulation 2                |
|--------------|-------------------------------|-----------------------------|
| Input layer  | 1 cell:1 unit (44 units)      | 1 cell:1 unit (44 units)    |
| Hidden layer | 1 cell:2 units (88 units)     | 1 cell:2 unit (88 units)    |
| Output layer | 1 cell : 1 unit (44 total units) | 11 main elevator MNs     |

The hidden units were recurrently connected and the interconnections between units were based on a 1st order exponential rise and decay. The general architecture has been described elsewhere (Faller and Luttges 1990).

For the cell-to-cell simulation no bias units were utilized. Since the MNs fire both synchronously and infrequently bias units were incorporated in the MN simulation. These units were constrained to function synchronously at the MN firing frequency. This constraining technique permitted the network to be trained despite the sparsity of the MN data set.

Training was performed using a supervised backpropagation algorithm in time. All 44 cells, 2000 points per discretized gradient ($\Delta=1$ msec real-time) were presented synchronously to the network. The results were consistent for $\Delta=2\text{-}5$ msec in all cases. The simulation paradigms were as follows:

|               | Simulation 1                | Simulation 2              |
|---------------|-----------------------------|---------------------------|
| Input         | Neural activity at time (t)  | Neural activity at time (t) |
| Output/Target | Neural activity at time (t+$\Delta$) | MN activity at time (t) |

Initial weights were random, -0.3 and 0.3, and the learning rate was $\eta=0.2$. Training was performed until the temporal reproduction of cell spiking patterns was verified for all cells. Following training, the network was "run", $\eta = 0$.

Sum squared errors for all units were calculated and normalized to an activation value of 0 to 1. The temporal reproduction of the output patterns was verified by linear correlation against the targeted spike trains. The "effective" contribution of each unit to the flight pattern was then determined by "lesioning" individual cells from the network prior to presenting the input pattern. The effects of lesioning were judged by the change in error relative to the unlesioned network.

# 3 RESULTS

## 3.1 CELL-TO-CELL SIMULATION

Following training the complete pattern set was presented to the network. And, the sum squared error was averaged over all units, Fig. 1. Clearly the network has a different "interpretation" of the data at certain time steps. This is due both to the omission/commission of spikes as well as timing errors. However, the data needed to reproduce overall cell firing patterns is clearly available.

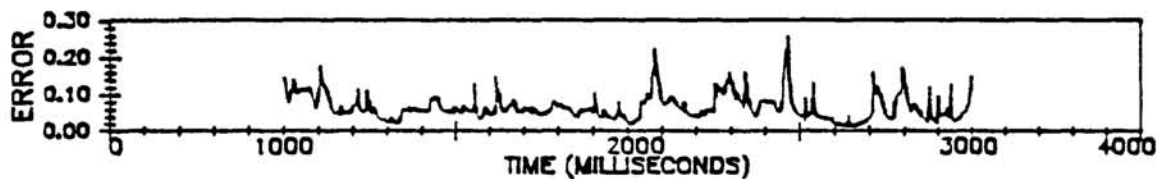

Figure 1: The network error

Unit sum squared errors were also averaged over the 2 second simulation, Fig. 2. Clearly the network predicted some unit/cell firing patterns easier than others.

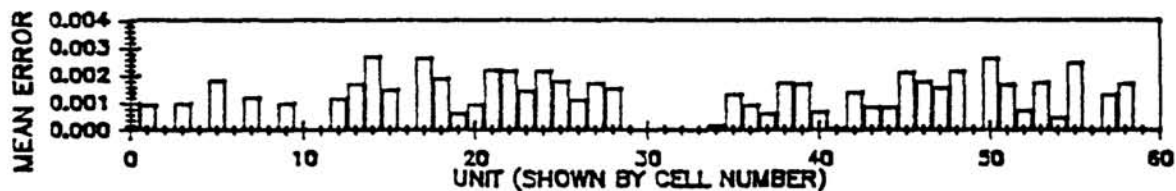

Figure 2: The unit errors

The temporal reproduction of the cell firing patterns was verified by linear correlation between the network outputs and the biological spike train representations. If the network accurately reproduces the temporal history of the spike trains these functions should be identical, r=1, Fig. 3. Clearly the network reproduces the temporal coding inherent within each spike train. The lowest correlation of roughly 0.85 is highly significant, (p<0.01).

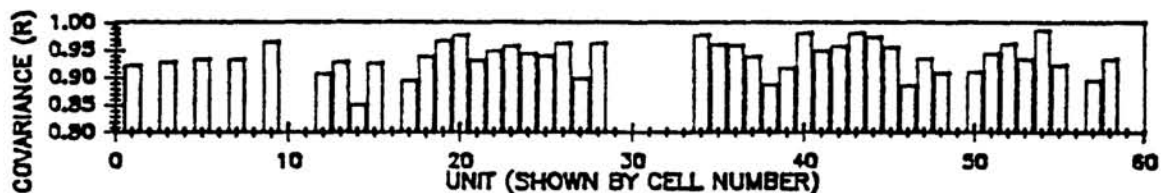

Figure 3: The unit temporal errors

One way to measure the relative importance of each unit/cell to the network is to omit/"lesion" each unit prior to presenting the cell firing patterns to the trained network. The data shown was collected by lesioning each unit individually, Fig. 4. The unlesioned network error is shown as the "0" cell. Overall the degradation of the network was minimal. Clearly some units provide more information to the network in reproducing the cell firing histories. Units that caused relatively large errors when "lesioned" were defined as primary units. The other units were defined as secondary units.

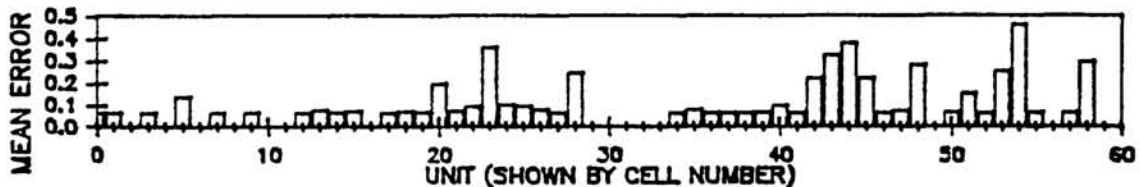

Figure 4: Lesion studies

The primary units (cells) form what might classically be termed a central pattern generator. These units can provide a relatively gross representation of both cellular and MN firing patterns. The generation of dynamic cellular and MN firing patterns, however, is apparently dependent on both primary and secondary units. It appears that the generation of functional activity patterns within the ganglia is largely controlled by the dynamic interactions between large groups of cells, ie. the "whole" network. This is consistent with other results derived from both neural network and statistical analyses of the biological data (Faller and Luttges 1990 & submitted:b).

## 3.2 MOTOR NEURON FIRING PATTERNS

The 44 cellular firing patterns were then used to drive the MN firing patterns. Following training, the cell firing pattern set was presented to the network and the sum squared error was averaged over the output MNs, Fig. 5. The error in this case oscillates in time at the wingbeat frequency of 37 Hz. As will be shown, however, this is an artifact and the network does accurately drive the MNs.

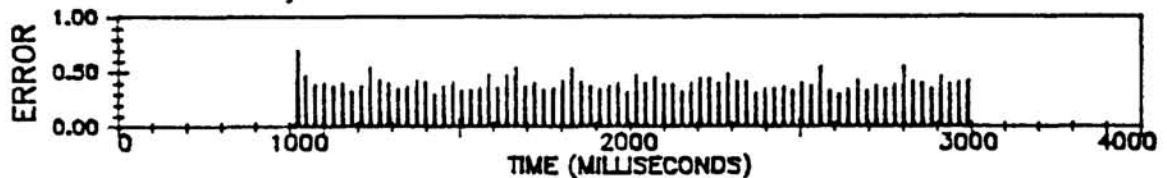

Figure 5: The network error

For each MN the sum squared error was also averaged over the 2 second simulation, Fig. 6. Clearly individual MNs contribute nearly equally to the network error.

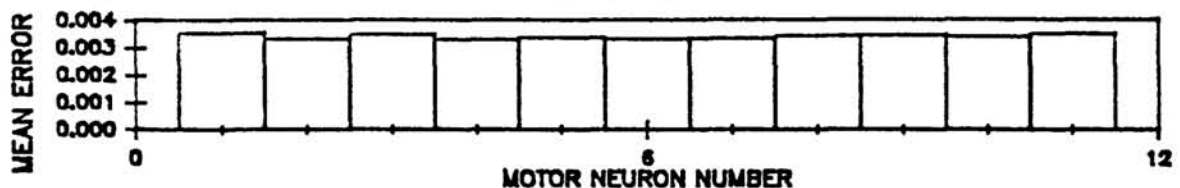

Figure 6: The unit errors

The temporal reproduction of the MN firing patterns was verified by linear correlation between the output and targeted MN firing patterns of the network. This is shown in Fig. 7. Clearly the cell inputs to the network have the spiking characteristics needed for driving the temporal firing sequences of the MNs innervating the wing musculature. All correlations are roughly 0.80, highly significant, ($p<0.01$). The output for one MN is shown relative to the targeted MN output in Fig. 8. Clearly the network does drive the MNs correctly.

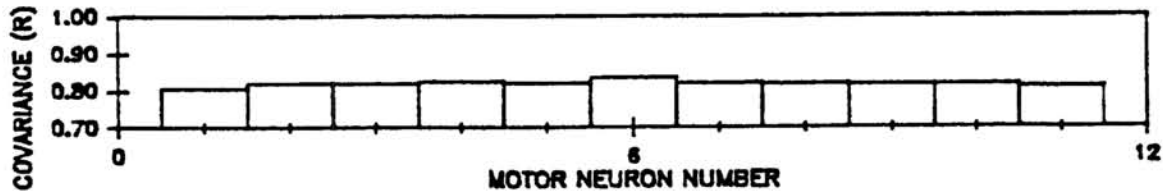

Figure 7: The unit temporal errors

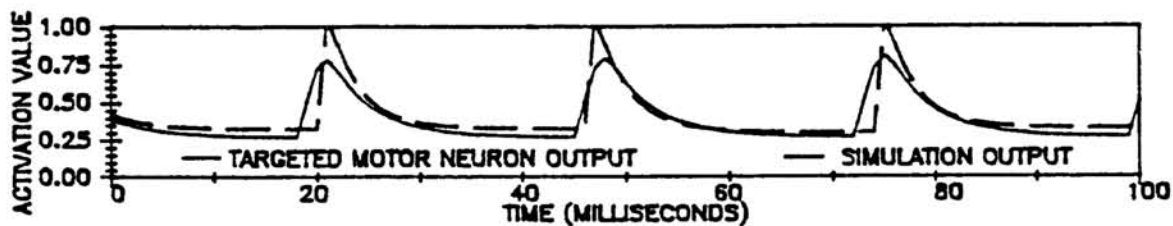

Figure 8: The MN firing patterns

## 3.3  SUMMARY

The results indicate that synthetic networks can learn and then synthesize patterns of neural spiking activity needed for biological function. In this case, cell and MN firing patterns occurring in the dragonfly ganglia during a spontaneous flight episode.

## 4  DISCUSSION

Recordings from more than 50 spatially unique cells that reflect the complex network characteristics of a small, intact neural tissue were used to successfully train two neural networks. Unit sum squared errors were less than 0.003 and spike train temporal histories were accurately reproduced. There was little evidence for unexpected "cellular behavior". Functional lesioning of single units in the network caused minimal degradation of network performance, however, some lesioned cells were more important than others to overall network performance.

The capability to lesion cells permitted the contribution of individual cells to the production of the flight rhythm to be determined. The detection of primary and secondary cells underlying the dynamic generation of both cellular and MN firing patterns is one example. Such results may encourage neurobiologists to adopt neural networks as effective analytical tools with which to study and analyze spike train data.

Clearly the solution arrived at is not the biological one. However, the networks do accurately predict the future cell firing patterns based on past firing history information. It

is asserted that the network must therefore contain the majority of information required to resolve biological cell interactions during flight in the dragonfly. A sample of 58 ganglionic cells was utilized, the remaining cells functional contributions are presumably statistically accounted for by this small sampling. The inherent "information" of the biological network is presumably stored in the weight matrices as a generalized statistical representation of the "rules" through which cells participate in biological assemblies.

Analyses of the weight matrices in turn may permit the operational "rules" of cell assemblies to be defined. Questions about the effects of cell size, the spatial architecture of the network and the temporal interactions between cells as they relate to cell assembly function can be addressed. For this reason the individuality of cells, the spatial architecture and the stochastic cellular firing histories of the individual cells were retained within the network architectures utilized. Crucial to these analyses will be methods that permit direct, time-incrementing evaluations of the weight matrices following training.

Biological nervous system function can now be analyzed from two points of view: direct analyses of the biological data and indirect, but potentially more approachable, analyses of the weight matrices from trained neural networks such as the ones described.

## REFERENCES

Faller WE, Luttges MW (1990) A Neural Network Simulation of Simultaneous Single-Unit Activity Recorded from the Dragonfly Ganglia. ISA Paper #90-033

Faller WE, Luttges MW (in press) Recording of Simultaneous Single-Unit Activity in the Dragonfly Ganglia. J Neurosci Methods

Faller WE, Luttges MW (Submitted:a) Spatiotemporal Analysis of Simultaneous Single-Unit Activity in the Dragonfly: I. Cellular Activity Patterns. Biol Cybern

Faller WE, Luttges MW (Submitted:b) Spatiotemporal Analysis of Simultaneous Single-Unit Activity in the Dragonfly: II. Network Connectivity. Biol Cybern

Hebb DO (1949) The Organization of Behavior: A Neuropsychological Theory. Wiley, New York, Chapman and Hall, London

Kliss MH (1989) Neurocontrol Systems and Wing-Fluid Interactions Underlying Dragonfly Flight. Ph.D. Thesis, University of Colorado, Boulder, pp 70-80

Luttges MW (1989) Accomplished Insect Fliers. In: Gad-el-Hak M (ed) Frontiers in Experimental Fluid Mechanics. Springer-Verlag, Berlin Heidelberg, pp 429-456

Simmons P (1977) The Neuronal Control of Dragonfly Flight I. Anatomy. J exp Biol 71:123-140

Somps C, Luttges MW (1985) Dragonfly flight: Novel uses of unsteady separated flows. Science 228:1326-1329


